# Application of SVMs for Colour Classification and Collision Detection with AIBO Robots

**Michael J. Quinlan, Stephan K. Chalup and Richard H. Middleton**$^{*}$
School of Electrical Engineering & Computer Science
The University of Newcastle, Callaghan 2308, Australia
{mquinlan,chalup,rick}@eecs.newcastle.edu.au

## Abstract

This article addresses the issues of colour classification and collision detection as they occur in the legged league robot soccer environment of RoboCup. We show how the method of one-class classification with support vector machines (SVMs) can be applied to solve these tasks satisfactorily using the limited hardware capacity of the prescribed Sony AIBO quadruped robots. The experimental evaluation shows an improvement over our previous methods of ellipse fitting for colour classification and the statistical approach used for collision detection.

## 1   Introduction

Autonomous agents offer a wide range of possibilities to apply and test machine learning algorithms, for example in vision, locomotion, and localisation. However, training-time requirements of sophisticated machine learning algorithms can overstrain the hardware of real world robots. Consequently, in most cases, ad hoc methods, hard coding of expert knowledge, and hand-tuning of parameters, or similar approaches were preferred over the use of learning algorithms on the robot. Application of the latter was often restricted to simulations which sometimes could support training or tuning of the real world robot parameters. However, often the gap between simulation and the real world was too wide so that a transfer of training results from the simulated to the real robot turned out to be useless.

A few years ago it may have been regarded as infeasible to consider the use of support vector machines [1, 2, 3] on real world robots with restricted processing capabilities. During the first years after their invention support vector machines had the reputation to be more a theoretical concept than a method which could be efficiently applied in real world situations. One of the main reasons for this was complexity of the quadratic programming part. In recent years it has become possible to speed up optimisations for SVMs in various ways [4]. SVMs have since been successfully applied on many tasks but primarily in the areas of data mining and pattern classification.

With the present study we explore the feasibility and usefulness of one-class SVM classification [5] for tasks faced by AIBO robots within the legged league environment of RoboCup [6]. We focus on two particularly critical issues: detection of objects based on

---

$^{*}$http://www.robots.newcastle.edu.au

correct colour classification and detection of robot-to-robot collisions. Both issues seemed not to be sufficiently solved and implemented by the teams of RoboCup2002 and caused significant deterioration in the quality of play even in the world-best teams of that league.

The article has five more sections addressing the environment and tasks, the methods, followed by the experiments and applications for colour classification and collision detection, respectively. The article concludes with a summary.

## 2    Environment and tasks

The restricted real world environment and the uniformly prescribed hardware of the legged league [6] of RoboCup provide a good compromise for testing machine learning algorithms on autonomous agents with a view towards possible applications in more general real world environments.

A soccer team in the legged league consists of four robots, including one goal keeper. Each team is identified by robots wearing either a red or blue coloured 'uniform'. The soccer matches take place on a green enclosed carpeted field with white boundaries. Two goals, a blue and a yellow, are positioned on opposite ends of the field. To aid localisation six beacons are placed regularly around the field, each uniquely identifiable by a specific colour pattern. The ball used is orange plastic and of a suitable size to be easily moved around by the robots. The games consist of two ten minute halves under strict rules imposed by independent referees.

The legged league of RoboCup 2003 prescribed the use of Sony AIBO entertainment robots, models ERS-210 or the newer ERS-210A. Both have an internal 64-bit RISC processor with clock speeds of 192MHz and 384MHz, respectively. The robots are programmed in a C++ software environment using the Sony's OPEN-R software development kit [7]. They have 16MB of memory accessible by user programs. The dimensions of the robot (width $\times$ height $\times$ length) are 154 mm $\times$ 266 mm $\times$ 274 mm (not including the tail and ears) and the mass is approximately 1.4 kg. The AIBO has 20 degrees of freedom (DOF): neck 3DOF (pan, tilt, and roll), ear 1DOF x 2, chin 1DOF, legs 3DOF (abductor, rotator, knee) x 4 and tail 2DOF (up-down, left-right).

Among other sensors the AIBO has a 1/6 inch colour CMOS camera capable of 25 frames per seconds. The images are gathered at a resolution of 352(H) $\times$ 288(V) but middleware restricts the available resolution to a maximum of 176(H) $\times$ 144(V). The lens has an aperture of 2.0 and a focal length of 2.18 mm. Additionally, the camera has a field of vision of $23.9°$ up and down and $28.8°$ left and right. To help achieve results in different lighting conditions the camera allows the modification of parameters: *White balance*, *Shutter Speed* and *Gain*.

### 2.1    Colour classification task

The vision system for most teams consists of four main tasks, *Colour Classification*, *Run Length Encoding*, *Blob Formation* and *Object Recognition* (Figure 1).

The classification process takes the image from the camera in a YUV bitmap format [8]. Each pixel in the image is assigned a colour label (i.e. ball orange, beacon pink etc.) based on its YUV values. A lookup table (LUT) is used to determine which YUV values correspond to which colour labels. The critical point is the initial generation of the LUT. Since the robot is extremely reliant on colour for object detection a new LUT has to be generated with any change in lighting conditions. Currently this is a manual task which requires a human to take hundreds of images and assign a colour label on a pixel-by-pixel basis. Using this method each LUT can take hours to create, yet it will still contain holes and classification errors.

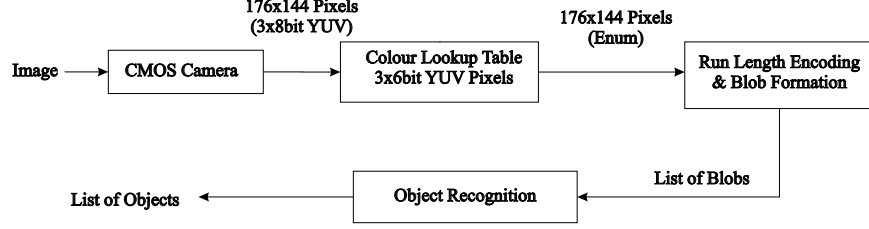

Figure 1: Vision System of the NUbots Legged League Team [9]

## 2.2 Collision detection task

The goal is to detect collisions using the limited sensors provided by the AIBO robot. The camera and infrared distance sensor on the AIBO don't provide enough support in avoiding obstacles unless the speed of the robot is dramatically decreased. For these reasons we have chosen to use information obtained from the joint sensors (i.e. the angle of the joint) as the input to our collision detection system [10].

## 3   One-class SVM classification method

An approach to one-class SVM classification was proposed by Schölkopf *et al.* [5]. Their strategy is to map data into the feature space corresponding to the kernel function and to separate them from the origin with maximum margin. This implies the construction of a hyperplane such that $w \cdot \Phi(x_i) - p \geq 0$. The result is a function $f$ that returns the value +1 in the region containing most of the data points and -1 elsewhere. Assuming the use of an RBF kernel and $i, j \in 1, ..., \ell$, we are presented with the dual problem:

$$\min_{\alpha} \frac{1}{2} \sum_{ij} \alpha_i \alpha_j k(x_i, x_j) \text{ subject to } 0 \leq \alpha_i \leq \frac{1}{\nu\ell} \, , \quad \sum_i \alpha_i = 1 \qquad (1)$$

$p$ can be found by the fact that for any such $\alpha_i$, a corresponding pattern $x_i$ satisfies:

$$p = \sum_j \alpha_j k(x_j, x_i)$$

The resulting decision function $f$ (the support of the distribution) is:

$$f(x) = sign(\sum_i \alpha_i k(x_i, x) - p)$$

An implementation of this approach is available in the LIBSVM library [11]. It solves a scaled version of (1):

$$\min_{\alpha} \frac{1}{2} \sum_{ij} \alpha_i \alpha_j k(x_i, x_j) \text{ subject to } 0 \leq \alpha_i \leq 1 \, , \quad \sum_i \alpha_i = \nu\ell$$

For our applications we use a RBF kernel with parameter $\gamma$ in the form $k(x, y) = e^{-\gamma\|x-y\|^2}$. The parameter $\nu$ approximates the fraction of outliers and support vectors [5].

## 3.1   Method for colour classification

The classification functions we seek take data that has been manually clustered to produce sets $X^k = \left\{ x_i^k \in \mathbf{R}^3; \ i = 1, ..., N_k \right\}$ of colour space data for each object colour $k$. Each

$X^k$ corresponds to sets of colour values in the YUV space corresponding to one of the known colour labels.

An individual one-class SVM is created for each colour, with $X^k$ being used as the training data (each element in the set is scaled between -1 and 1). By training with an extremely low $\nu$ and a large $\gamma$ the boundary formed by the decision function approximates the region that contains the majority (1-$\nu$) of the points in $X^k$. In addition the SVM has the advantage of simultaneously removing the outliers that occur during manual classification.

The new colour set is constructed by attempting to classify every point in the YUV space ($64^3$ elements). All points that return a value of +1 are inside the region and therefore deemed to be of colour $k$.

One-class SVM was chosen because it allows us to optimally treat each individual colour. To avoid misclassification each point in YUV space that does not strongly correspond to one of the known colours must remain classified as unknown. In addition the colours were originally selected because they are located in different areas of the YUV space. Because of this we can choose to treat each colour without regard to the location and shape of the other colours. For these reasons we are not interested in using a multi-class technique to form a hyperplane that provides an optimal separation between the colours.

### 3.2 Method for collision detection

For collision detection the one-class SVM is employed as a novelty detection mechanism. In our implementation each training point is a vector containing thirteen elements. These include five walk parameters, *stepFrequency*, *backStrideLength*, *turn*, *strafe* and *timeParameter* along with a sensor reading from the abductor and rotator joints on each of the four legs. Upon training the SVMs decision function will return +1 for all values that relate to a "normal" step, and -1 for all steps that contain a fault.

Speed is of the greatest importance in the Robocup domain. For this reason a collision detection system must attempt to minimise the generation of false-positives (detecting a collision that we deemed not to have happened) while still finding a high percentage of actual collisions. Low false-positives are achieved by keeping the kernel parameter $\gamma$ high but this has the side effect of lowering the generalisation to the data set, which results in the need for an increased number of training points. In a real world robotic system the need for more training points greatly increases the training time and in-turn the wear on the machinery.

## 4 Experiments and application to colour classification

The SVM can be used in two situations during the colour classification procedure. Firstly during the construction of a new LUT where it can be applied to increase the speed of classification.

By lowering $\gamma$ while the number of training points is low, a rough estimation of the final shape can be obtained. By continuing the manual classification and increasing $\gamma$ a closer approximation to the area containing the training data is obtained. In this manner a continually improving LUT can be constructed until it is deemed adequate.

An extreme example of this application is during the set-up phase at a competition. In the past when we arrived at a new venue *all* system testing was delayed until the generation of a LUT. Of critical importance is testing the locomotion engine on the new carpet and in particular ball chasing. The task of ball chasing relies on the classification of ball orange. Thus a method of quickly but roughly classifying orange is valuable. By manually classifying a few images of the ball and then training the SVM with $\gamma < 0$, a sphere containing

all possible values for the ball is generated.

The second situation in which we use the one-class SVM is on a completed LUT. Either all colours in the table can be trained (i.e. updating of an old table) or an individual colour is trained due to an initial classification error. This procedure can be performed either on the robot or a remote computer.

Empirical tests have indicated that $\nu = 0.025$ and $\gamma = 250$ provide excellent results on a previously constructed LUT. The initial table contained 3329 entries while after training the table contained 6989 entries. The most evident change can be seen in the classification of colour white, see Figure 2.

The LUTs were compared over 60 images, which equates to 1,520,640 individual pixel comparisons. The initial table generated 144,098 classification errors. The new LUT produced 117,652 errors, this equates to an 18% reduction in errors.

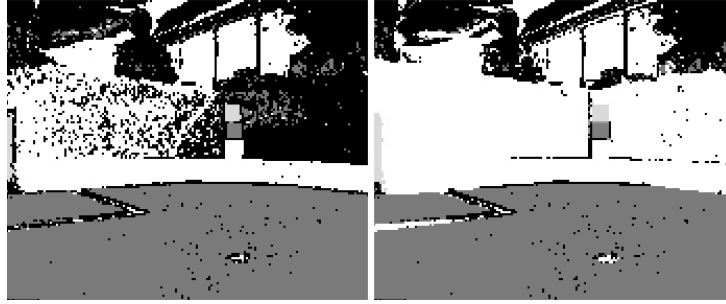

Figure 2: Image Comparison: The left image is classified with the original LUT and the image on the right is the using the updated LUT. Black pixels indicate an unknown colour.

### 4.1 Comparison with ellipsoid fitting

The previous method involved converting the existing LUT values from YUV to the HSI colour space [8] and fitting an ellipsoid, $E$, which can be represented by the quadratic form:

$$E(x_0, Q) = \left\{ x \in \mathbf{R}^3 : (x - x_0)^T Q^{-1} (x - x_0) \leq 1 \right\} \qquad (2)$$

where $x_0$ is the centre of the ellipsoid, and the size, orientation and shape of the ellipsoid are contained in the positive definite symmetric matrix $Q = Q^T > 0 \in \mathbf{R}^{3 \times 3}$.

Note that this definition of the shape can be alternatively represented by the linear matrix inequality (LMI):

$$x_i \in E = \begin{bmatrix} Q & (x_i - x_0) \\ (x_i - x_0)^T & 1 \end{bmatrix} \geq 0 \qquad (3)$$

The LMI (3) is linear in the unknowns $Q$ and $x_0$ and this therefore leads to the convex optimisation:

$$(Q, x_0) = \underset{\substack{Q = Q^T > 0, x_0 : \\ (3) \text{ is true for } i = 1..N_k}}{argmin} \{tr(Q)\}$$

Note that minimising the trace of $\mathbf{Q}$ ($tr(Q)$) is the same as minimising the sum of the diagonal elements of $\mathbf{Q}$ which is the same as minimising the sum of the squares of the

lengths of the principal axes of the ellipsoid. The ellipsoidal shape defined in (2) has the disadvantage of restricting the shape of possible regions in the colour space. However, it does have the advantage of having a simple representation and a convex shape.

Before the ellipsoid can be fitted, potential outliers and duplicate points were identified and removed. The removal of outliers is important in avoiding too large a region. Duplicate points were removed, since these increase computations without adding any information.

For the comparison we use the initial LUT from the above example. Figure 3 shows the effects of each method on the colour white. To make the comparison with ellipsoids, the initial LUT and the generated LUT from the SVM procedure are shown in the HSI colour space.

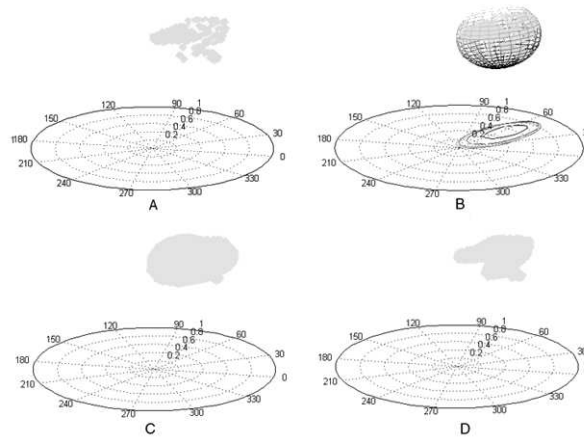

Figure 3: Colour classification in HSI colour space: A) Points manually classified at white. B) Ellipsoid fitted to these white points. C) Result of the one-class SVM technique, $\nu$=0.025 and $\gamma$=10. D) Result of the one-class SVM technique, $\nu$=0.025 and $\gamma$=250.

It is evident that the manual classification of white is rather incomplete and contains many holes that should be classified as white. The negative results of these holes can be seen as noise in the left image of Figure 2.

Using the ellipsoid fitting method these holes are filled but with the potential drawback of over classification. From image B in Figure 3 it is evident that the top section and the bottom left of the ellipsoid contain no white entries and therefore it is highly questionable that this area should be classified as white.

Images C and D in the figure show the results of our one-class SVM method. It is clear from image D that the area now classified as white is a region that tightly fits the original training set.

## 5   Experiments and application to collision detection

The collision detection system is designed with the aim that the entire system can be run on the robot. This means adhering to the memory and processing capabilities of the device. On the AIBO we have a maximum of 8MB memory available for collision detection, a total of

20,000 training points. This is the equivalent of 1000 steps which equates to approximately 10 minutes of training time. The training set is generated by having the robot behave normally on the field but with the stipulation that all collisions are avoided.

The trained classifier analyses the on-line stream of joint data measurements in samples of ten consecutive data points. If more than 2 points in one sample are classified as -1 a collision is declared to be detected.

Initial parameters of $\nu = 0.05$ and $\gamma = 5$ were chosen, this was based on the assumption that a collision point would lie considerably outside the training set. The results from these parameters were less then satisfying, only the largest of collisions (i.e. physically holding multiple legs) were detected. The solution to this problem could involve increasing $\nu$ due to the possibility that the initial training set contained many outliers and/or increasing $\gamma$ to improve the tightness of the classification.

By a series of tests, all of which tended to lead to either an over classification or an under classification, parameters of $\nu = 0.05$ and $\gamma = 100$ were settled on. In our system these parameters appear to give the best balance between minimising false-positives and maximising correct detection of collisions.

## 5.1 Comparison with the previous statistical method

The previous method, described in [10], for collision detection involves observing a joint position substantially differing from its expected value. In our case an empirical study found two standard deviations to be a practical measure, see Figure 4. Initially we would have considered a collision to have occurred if a single error is found, but further investigation has shown that finding multiple errors (in most cases three) in quick succession is necessary to warrant a warning that can be acted upon by the robot's behaviour system.

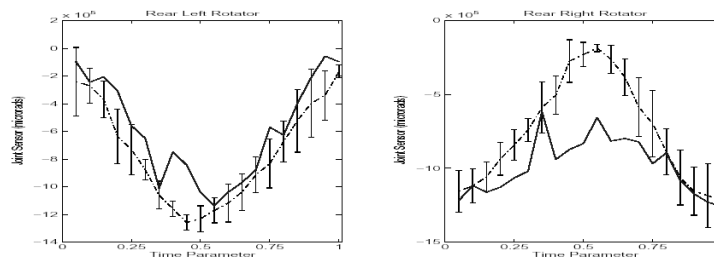

Figure 4: Rear Rotators for a forwards walking boundary collision on both front legs, front right leg hitting first. The bold line shows the path of a collided motion. The dotted line represents the mean "normal" path of the joint (that is, during unobstructed motion), with the error bars indicating two standard deviations above and below.

One drawback of this method is that it relied on domain knowledge to arrive at two standard deviations. In addition it required considerable storage space to hold the table of means and standard deviations for each parameter combination.

The previous statistical method had the advantage of extremely low computational expense, in fact it was a table look up. The trade-off is increased space, this method required the allocation of approximately 6MB of memory during both the training and detection stages. Conversely the SVM approach requires only about 1MB of memory during the detection phase, but this comes at the side effect of increased computation. Since the SVM approach was capable of running without reducing the frame rate, the extra memory could now be used for other applications.

With respect to accuracy the SVM approach slightly outperformed the original statistical method for particular types of steps, these include the common steps associated with chasing the ball. Other step types, such as an aggressive turn did not show the same improvement. This is due to the movement of the joints in some motions being more inconsistent, thus making accurate classification harder.

A possible solution may involve using multiple SVMs associated with different combinations of walk parameters, allowing the tuning of parameters on a specific basis. This solution would have the downside of requiring more memory.

## 6  Summary

The method of one-class classification with SVMs was successfully applied to the tasks of colour classification and collision detection using the restricted memory and processing power of the AIBO hardware. It was possible to run the SVM algorithm implemented in the C++ libraries of LIBSVM off and on the robot. In a comparison with previously used methods the SVM based methods generated better results, and in the case of colour classification the SVM approach was more efficient and convenient.

**Acknowledgments**

We would like to thank William McMahan and Jared Bunting for their work on the previous vision classification method and Craig Murch for his extensive contributions to both the vision and locomotion systems. Michael J. Quinlan was supported by a University of Newcastle Postgraduate Research Scholarship.

**References**

[1] B. E. Boser, I. M. Guyon, and V. N. Vapnik. A training algorithm for optimal margin classifiers. In D. Haussler, editor, *Proceedings of the 5th Annual ACM Workshop on Computational Learning Theory*, pages 144–152, Pittsburgh, PA, July 1992. ACM Press.

[2] C. Cortes and V. Vapnik. Support vector networks. *Machine Learning*, 20:273 – 297, 1995.

[3] V. Vapnik. *The Nature of Statistical Learning Theory*. Springer Verlag, New York, 1995.

[4] Bernhard Schölkopf and Alexander J. Smola. *Learning with Kernels, Support Vector Machines, Regularization, Optimization and Beyond*. The MIT Press, 2002.

[5] B. Schölkopf, J. C. Platt, J. Shawe-Taylor, A. J. Smola, and R. C. Williamson. Estimating the support of a high-dimensional distribution. *Neural Computation*, 13:1443–1471, 2001.

[6] RoboCup Legged League web site. http://www.openr.org/robocup/index.html.

[7] OPEN-R SDK. http://openr.aibo.com.

[8] Linda G. Shapiro and George C. Stockman. *Computer Vision*. Prentice Hall, 2001.

[9] J. Bunting, S. Chalup, M. Freeston, W. McMahan, R. Middleton, C. Murch, M. Quinlan, C. Seysener, and G. Shanks. Return of the NUbots! The 2003 NUbots Team Report, 2003. http://robots.newcastle.edu.au/publications/NUbotFinalReport2003.pdf.

[10] Michael J. Quinlan, Craig L. Murch, Richard H. Middleton, and Stephan K. Chalup. Traction monitoring for collision detection with legged robots. In *RoboCup 2003 Symposium*, 2003.

[11] Chih-Chung Chang and Chih-Jen Lin. *LIBSVM: a library for support vector machines*, 2001. Software available at http://www.csie.ntu.edu.tw/~cjlin/libsvm.
